# Variational Bayesian Stochastic Complexity of Mixture Models

**Kazuho Watanabe**[*]
Department of Computational Intelligence
and Systems Science
Tokyo Institute of Technology
Mail Box:R2-5, 4259 Nagatsuta,
Midori-ku, Yokohama, 226-8503, Japan
kazuho23@pi.titech.ac.jp

**Sumio Watanabe**
P& I Lab.
Tokyo Institute of Technology
swatanab@pi.titech.ac.jp

## Abstract

The Variational Bayesian framework has been widely used to approximate the Bayesian learning. In various applications, it has provided computational tractability and good generalization performance. In this paper, we discuss the Variational Bayesian learning of the mixture of exponential families and provide some additional theoretical support by deriving the asymptotic form of the stochastic complexity. The stochastic complexity, which corresponds to the minimum free energy and a lower bound of the marginal likelihood, is a key quantity for model selection. It also enables us to discuss the effect of hyperparameters and the accuracy of the Variational Bayesian approach as an approximation of the true Bayesian learning.

## 1  Introduction

The Variational Bayesian (VB) framework has been widely used as an approximation of the Bayesian learning for models involving hidden (latent) variables such as mixture models[2][4]. This framework provides computationally tractable posterior distributions with only modest computational costs in contrast to Markov chain Monte Carlo (MCMC) methods. In many applications, it has performed better generalization compared to the maximum likelihood estimation.

In spite of its tractability and its wide range of applications, little has been done to investigate the theoretical properties of the Variational Bayesian learning itself. For example, questions like how accurately it approximates the true one remained unanswered until quite recently. To address these issues, the stochastic complexity in the Variational Bayesian learning of gaussian mixture models was clarified and the accuracy of the Variational Bayesian learning was discussed[10].

[*]This work was supported by the Ministry of Education, Science, Sports and Culture, Grant-in-Aid for JSPS Fellows 4637 and for Scientific Research 15500130, 2005.

In this paper, we focus on the Variational Bayesian learning of more general mixture models, namely the mixtures of exponential families which include mixtures of distributions such as gaussian, binomial and gamma. Mixture models are known to be non-regular statistical models due to the non-identifiability of parameters caused by their hidden variables[7]. In some recent studies, the Bayesian stochastic complexities of non-regular models have been clarified and it has been proven that they become smaller than those of regular models[12][13]. This indicates an advantage of the Bayesian learning when it is applied to non-regular models.

As our main results, the asymptotic upper and lower bounds are obtained for the stochastic complexity or the free energy in the Variational Bayesian learning of the mixture of exponential families. The stochastic complexity is important quantity for model selection and giving the asymptotic form of it also contributes to the following two issues. One is the accuracy of the Variational Bayesian learning as an approximation method since the stochastic complexity shows the distance from the variational posterior distribution to the true Bayesian posterior distribution in the sense of Kullback information. Indeed, we give the asymptotic form of the stochastic complexity as $\overline{F}(n) \simeq \overline{\lambda} \log n$ where $n$ is the sample size, by comparing the coefficient $\overline{\lambda}$ with that of the true Bayesian learning, we discuss the accuracy of the VB approach. Another is the influence of the hyperparameter on the learning process. Since the Variational Bayesian algorithm is a procedure of minimizing the functional that finally gives the stochastic complexity, the derived bounds indicate how the hyperparameters influence the process of the learning. Our results have an implication for how to determine the hyperparameter values before the learning process.

We consider the case in which the true distribution is contained in the learner model. Analyzing the stochastic complexity in this case is most valuable for comparing the Variational Bayesian learning with the true Bayesian learning. This is because the advantage of the Bayesian learning is typical in this case[12]. Furthermore, this analysis is necessary and essential for addressing the model selection problem and hypothesis testing.

The paper is organized as follows. In Section 2, we introduce the mixture of exponential family model. In Section 3, we describe the Bayesian learning. In Section 4, the Variational Bayesian framework is described and the variational posterior distribution for the mixture of exponential family model is derived. In Section 5, we present our main result. Discussion and conclusion follow in Section 6.

## 2 Mixture of Exponential Family

Denote by $c(x|b)$ a density function of the input $x \in R^N$ given an $M$-dimensional parameter vector $b = (b^{(1)}, b^{(2)}, \cdots, b^{(M)})^T \in B$ where $B$ is a subset of $R^M$. The general mixture model $p(x|\theta)$ with a parameter vector $\theta$ is defined by

$$p(x|\theta) = \sum_{k=1}^{K} a_k c(x|b_k),$$

where integer $K$ is the number of components and $\{a_k | a_k \geq 0, \sum_{k=1}^{K} a_k = 1\}$ is the set of mixing proportions. The model parameter $\theta$ is $\{a_k, b_k\}_{k=1}^{K}$.

A mixture model is called a mixture of exponential family (MEF) model or exponential family mixture model if the probability distribution $c(x|b)$ for each component is given by the following form,

$$c(x|b) = \exp\{b \cdot f(x) + f_0(x) - g(b)\}, \tag{1}$$

where $b \in B$ is called the natural parameter, $b \cdot f(x)$ is its inner product with the vector $f(x) = (f_1(x), \cdots, f_M(x))^T$, $f_0(x)$ and $g(b)$ are real-valued functions of the input $x$ and the parameter $b$, respectively[3]. Suppose functions $f_1, \cdots, f_M$ and a constant function are linearly independent, which means the effective number of parameters in a single component distribution $c(x|b)$ is $M$.

The conjugate prior distribution $\varphi(\theta)$ for the MEF model is given by the product of the following two distributions on $\mathbf{a} = \{a_k\}_{k=1}^K$ and $\mathbf{b} = \{b_k\}_{k=1}^K$,

$$\varphi(\mathbf{a}) = \frac{\Gamma(K\phi_0)}{\Gamma(\phi_0)^K} \prod_{k=1}^K a_k^{\phi_0-1}, \tag{2}$$

$$\varphi(\mathbf{b}) = \prod_{k=1}^K \varphi(b_k) = \prod_{k=1}^K \frac{\exp\{\xi_0(b_k \cdot \nu_0 - g(b_k))\}}{C(\xi_0, \nu_0)}, \tag{3}$$

where $\xi_0 > 0$, $\nu_0 \in R^M$ and $\phi_0 > 0$ are constants called hyperparameters and

$$C(\xi, \mu) = \int \exp\{\xi(\mu \cdot b - g(b))\}db \tag{4}$$

is a function of $\xi \in R$ and $\mu \in R^M$.

The mixture model can be rewritten as follows by using a hidden variable $y = (y^1, \cdots, y^K) \in \{(1, 0, \cdots, 0), (0, 1, \cdots, 0), \cdots, (0, 0, \cdots, 1)\}$,

$$p(x, y|\theta) = \prod_{k=1}^K \left[ a_k c(x|b_k) \right]^{y^k}.$$

If and only if the datum $x$ is generated from the $k$th component, $y^k = 1$.

## 3    The Bayesian Learning

Suppose $n$ training samples $X^n = \{x_1, \cdots, x_n\}$ are independently and identically taken from the true distribution $p_0(x)$. In the Bayesian learning of a model $p(x|\theta)$ whose parameter is $\theta$, first, the prior distribution $\varphi(\theta)$ on the parameter $\theta$ is set. Then the posterior distribution $p(\theta|X^n)$ is computed from the given dataset and the prior by

$$p(\theta|X^n) = \frac{1}{Z(X^n)} \exp(-nH_n(\theta))\varphi(\theta), \tag{5}$$

where $H_n(\theta)$ is the empirical Kullback information,

$$H_n(\theta) = \frac{1}{n} \sum_{i=1}^n \log \frac{p_0(x_i)}{p(x_i|\theta)}, \tag{6}$$

and $Z(X^n)$ is the normalization constant that is also known as the marginal likelihood or the evidence of the dataset $X^n$[6]. The Bayesian predictive distribution $p(x|X^n)$ is given by averaging the model over the posterior distribution as follows,

$$p(x|X^n) = \int p(x|\theta)p(\theta|X^n)d\theta. \tag{7}$$

The stochastic complexity $F(X^n)$ is defined by

$$F(X^n) = -\log Z(X^n), \tag{8}$$

which is also called the free energy and is important in most data modelling problems. Practically, it is used as a criterion by which the model is selected and the hyperparameters in the prior are optimized[1][9].

Define the average stochastic complexity $F(n)$ by

$$F(n) = E_{X^n}\big[F(X^n)\big], \tag{9}$$

where $E_{X^n}[\cdot]$ denotes the expectation value over all sets of training samples. Recently, it was proved that $F(n)$ has the following asymptotic form[12],

$$F(n) \simeq \lambda \log n - (m-1) \log \log n + O(1), \tag{10}$$

where $\lambda$ and $m$ are the rational number and the natural number respectively which are determined by the singularities of the set of true parameters. In regular statistical models, $2\lambda$ is equal to the number of parameters and $m = 1$, whereas in non-regular models such as mixture models, $2\lambda$ is not larger than the number of parameters and $m \geq 1$. This means an advantage of the Bayesian learning.

However, in the Bayesian learning, one computes the stochastic complexity or the predictive distribution by integrating over the posterior distribution, which typically cannot be performed analytically. As an approximation, the VB framework was proposed[2][4].

# 4 The Variational Bayesian Learning

## 4.1 The Variational Bayesian Framework

In the VB framework, the Bayesian posterior $p(Y^n, \theta|X^n)$ of the hidden variables and the parameters is approximated by the variational posterior $q(Y^n, \theta|X^n)$, which factorizes as

$$q(Y^n, \theta|X^n) = Q(Y^n|X^n)r(\theta|X^n), \tag{11}$$

where $Q(Y^n|X^n)$ and $r(\theta|X^n)$ are posteriors on the hidden variables and the parameters respectively. The variational posterior $q(Y^n, \theta|X^n)$ is chosen to minimize the functional $\overline{F}[q]$ defined by

$$\overline{F}[q] = \sum_{Y^n} \int q(Y^n, \theta|X^n) \log \frac{q(Y^n, \theta|X^n)p_0(X^n)}{p(X^n, Y^n, \theta)} d\theta, \tag{12}$$

$$= F(X^n) + K(q(Y^n, \theta|X^n)||p(Y^n, \theta|X^n)), \tag{13}$$

where $K(q(Y^n, \theta|X^n)||p(Y^n, \theta|X^n))$ is the Kullback information between the true Bayesian posterior $p(Y^n, \theta|X^n)$ and the variational posterior $q(Y^n, \theta|X^n)$ [1]. This leads to the following theorem. The proof is well known[8].

**Theorem 1** *If the functional $\overline{F}[q]$ is minimized under the constraint (11) then the variational posteriors, $r(\theta|X^n)$ and $Q(Y^n|X^n)$, satisfy*

$$r(\theta|X^n) = \frac{1}{C_r}\varphi(\theta) \exp \big\langle \log p(X^n, Y^n|\theta) \big\rangle_{Q(Y^n|X^n)}, \tag{14}$$

$$Q(Y^n|X^n) = \frac{1}{C_Q} \exp \big\langle \log p(X^n, Y^n|\theta) \big\rangle_{r(\theta|X^n)}, \tag{15}$$

$$K(q(x)||p(x)) = \int q(x) \log \frac{q(x)}{p(x)} dx.$$

where $C_r$ and $C_Q$ are the normalization constants[2].

We define the stochastic complexity in the VB learning $\overline{F}(X^n)$ by the minimum value of the functional $\overline{F}[q]$ , that is ,

$$\overline{F}(X^n) = \min_{r,Q} \overline{F}[q],$$

which shows the accuracy of the VB approach as an approximation of the Bayesian learning. $\overline{F}(X^n)$ is also used for model selection since it gives an upper bound of the true Bayesian stochastic complexity $F(X^n)$.

## 4.2   Variational Posterior for MEF Model

In this subsection, we derive the variational posterior $r(\theta|X^n)$ for the MEF model based on (14) and then define the variational parameter for this model.

Using the complete data $\{X^n, Y^n\} = \{(x_1, y_1), \cdots, (x_n, y_n)\}$, we put

$$\overline{y}_i^k = \langle y_i^k \rangle_{Q(Y^n)}, \quad n_k = \sum_{i=1}^n \overline{y}_i^k, \text{and} \quad \nu_k = \frac{1}{n_k} \sum_{i=1}^n \overline{y}_i^k f(x_i),$$

where $y_i^k = 1$ if and only if the $i$th datum $x_i$ is from the $k$th component. The variable $n_k$ is the expected number of the data that are estimated to be from the $k$th component. From (14) and the respective prior (2) and (3), the variational posterior $r(\theta)$ is obtained as the product of the following two distributions[3],

$$r(\mathbf{a}) = \frac{\Gamma(n + K\phi_0)}{\prod_{k=1}^K \Gamma(n_k + \phi_0)} \prod_{k=1}^K a_k^{n_k + \phi_0 - 1}, \tag{16}$$

$$r(\mathbf{b}) = \prod_{k=1}^K r(b_k) = \prod_{k=1}^K \frac{1}{C(\gamma_k, \overline{\mu}_k)} \exp\{\gamma_k(\overline{\mu}_k \cdot b_k - g(b_k))\}, \tag{17}$$

where $\overline{\mu}_k = \frac{n_k \nu_k + \xi_0 \nu_0}{n_k + \xi_0}$ and $\gamma_k = n_k + \xi_0$. Let

$$\overline{a}_k = \langle a_k \rangle_{r(\mathbf{a})} = \frac{n_k + \phi_0}{n + K\phi_0}, \tag{18}$$

$$\overline{b}_k = \langle b_k \rangle_{r(b_k)} = \frac{1}{\gamma_k} \frac{\partial \log C(\gamma_k, \overline{\mu}_k)}{\partial \overline{\mu}_k}, \tag{19}$$

and define the variational parameter $\overline{\theta}$ by $\overline{\theta} = \langle \theta \rangle_{r(\theta)} = \{\overline{a}_k, \overline{b}_k\}_{k=1}^K$. Then it is noted that the variational posterior $r(\theta)$ and $C_Q$ in (15) are parameterized by the variational parameter $\overline{\theta}$. Therefore, we denote them as $r(\theta|\overline{\theta})$ and $C_Q(\overline{\theta})$ henceforth. We define the variational estimator $\overline{\theta}_{vb}$ by the variational parameter $\overline{\theta}$ that attains the minimum value of the stochastic complexity $\overline{F}(X^n)$. Then, putting (15) into (12), we obtain

$$\overline{F}(X^n) = \min_{\overline{\theta}}\{K(r(\theta|\overline{\theta})||\varphi(\theta)) - (\log C_Q(\overline{\theta}) + S(X^n))\}, \tag{20}$$

$$= K(r(\theta|\overline{\theta}_{vb})||\varphi(\theta)) - (\log C_Q(\overline{\theta}_{vb}) + S(X^n)), \tag{21}$$

where $S(X^n) = -\sum_{i=1}^n \log p_0(x)$.

Therefore, our aim is to evaluate the minimum value of (20) as a function of the variational parameter $\overline{\theta}$.

## 5 Main Result

The average stochastic complexity $\overline{F}(n)$ in the VB learning is defined by

$$\overline{F}(n) = E_{X^n}[\overline{F}(X^n)]. \tag{22}$$

We assume the following conditions.

(i) The true distribution $p_0(x)$ is an MEF model $p(x|\theta_0)$ which has $K_0$ components and the parameter $\theta_0 = \{a_k^*, b_k^*\}_{k=1}^{K_0}$,

$$p(x|\theta_0) = \sum_{k=1}^{K_0} a_k^* \exp\{b_k^* \cdot f(x) + f_0(x) - g(b_k^*)\},$$

where $b_k^* \in R^M$ and $b_k^* \neq b_j^*(k \neq j)$. And suppose that the model $p(x|\theta)$ has $K$ components,

$$p(x|\theta) = \sum_{k=1}^{K} a_k \exp\{b_k \cdot f(x) + f_0(x) - g(b_k)\},$$

and $K \geq K_0$ holds.

(ii) The prior distribution of the parameters is $\varphi(\theta) = \varphi(\mathbf{a})\varphi(\mathbf{b})$ given by (2) and (3) with $\varphi(\mathbf{b})$ bounded.

(iii) Regarding the distribution $c(x|b)$ of each component, the Fisher information matrix $I(b) = \frac{\partial^2 g(b)}{\partial b \partial b}$ satisfies $0 < |I(b)| < +\infty$, for arbitrary $b \in B$ [4]. The function $\mu \cdot b - g(b)$ has a stationary point at $\hat{b}$ in the interior of $B$ for each $\mu \in \{\frac{\partial g(b)}{\partial b}|b \in B\}$.

Under these conditions, we prove the following.

**Theorem 2 (Main Result)** *Assume the conditions (i),(ii) and (iii). Then the average stochastic complexity $\overline{F}(n)$ defined by (22) satisfies*

$$\underline{\lambda}\log n + E_{X^n}\left[nH_n(\overline{\theta}_{vb})\right] + C_1 \leq \overline{F}(n) \leq \overline{\lambda}\log n + C_2, \tag{23}$$

*for an arbitrary natural number $n$, where $C_1, C_2$ are constants independent of $n$ and*

$$\underline{\lambda} = \left\{ \begin{array}{l} (K-1)\phi_0 + \frac{M}{2}, \\ \frac{MK+K-1}{2}, \end{array} \right. \quad \overline{\lambda} = \left\{ \begin{array}{ll} (K-K_0)\phi_0 + \frac{MK_0+K_0-1}{2} & (\phi_0 \leq \frac{M+1}{2}), \\ \frac{MK+K-1}{2} & (\phi_0 > \frac{M+1}{2}). \end{array} \right. \tag{24}$$

This theorem shows the asymptotic form of the average stochastic complexity in the Variational Bayesian learning. The coefficients $\underline{\lambda}$, $\overline{\lambda}$ of the leading terms are identified by $K, K_0$, that are the numbers of components of the learner and the true distribution, the number of parameters $M$ of each component and the hyperparameter $\phi_0$ of the conjugate prior given by (2).

In this theorem, $nH_n(\overline{\theta}_{vb}) = -\sum_{i=1}^{n} \log p(x_i|\overline{\theta}_{vb}) - S(X^n)$, and $-\sum_{i=1}^{n} \log p(x_i|\overline{\theta}_{vb})$ is a training error which is computable during the learning. If the term $E_{X^n}\left[nH_n(\overline{\theta}_{vb})\right]$ is a bounded function of $n$, then it immediately follows from this theorem that

$$\underline{\lambda}\log n + O(1) \leq \overline{F}_0(n) \leq \overline{\lambda}\log n + O(1),$$

where $O(1)$ is a bounded function of $n$. In certain cases, such as binomial mixtures and mixtures of von-Mises distributions, it is actually a bounded function of $n$. In the case of gaussian mixtures, if $B = R^N$, it is conjectured that the minus likelihood ratio $\min_\theta nH_n(\theta)$, a lower bound of $nH_n(\overline{\theta}_{vb})$, is at most of the order of $\log\log n$[5].

Since the dimension of the parameter $\theta$ is $MK + K - 1$, the average stochastic complexity of regular statistical models, which coincides with the Bayesian information criterion (BIC)[9] is given by $\lambda_{\mathrm{BIC}} \log n$ where $\lambda_{\mathrm{BIC}} = \frac{MK+K-1}{2}$. Theorem 2 claims that the coefficient $\overline{\lambda}$ of $\log n$ is smaller than $\lambda_{\mathrm{BIC}}$ when $\phi_0 \leq (M+1)/2$. This implies that the advantage of non-regular models in the Bayesian learning still remains in the VB learning.

**(Outline of the proof of Theorem 2)**

From the condition (iii), calculating $C(\gamma_k, \overline{\mu}_k)$ in (17) by the saddle point approximation, $K(r(\theta|\overline{\theta})||\varphi(\theta))$ in (20) is evaluated as follows [5],

$$K(r(\theta|\overline{\theta})||\varphi(\theta)) = G(\overline{\mathbf{a}}) - \sum_{k=1}^{K} \log \varphi(\overline{b}_k) + O_p(1), \tag{25}$$

where the function $G(\overline{\mathbf{a}})$ of $\overline{\mathbf{a}} = \{\overline{a}_k\}_{k=1}^{K}$ is given by

$$G(\overline{\mathbf{a}}) = \frac{MK + K - 1}{2} \log n + \{\frac{M}{2} - (\phi_0 - \frac{1}{2})\} \sum_{k=1}^{K} \log \overline{a}_k. \tag{26}$$

Then $\log C_Q(\overline{\theta})$ in (20) is evaluated as follows.

$$nH_n(\overline{\theta}) + O_p(1) \leq -(\log C_Q(\overline{\theta}) + S(X^n)) \leq n\overline{H}_n(\overline{\theta}) + O_p(1) \tag{27}$$

where

$$\overline{H}_n(\overline{\theta}) = \frac{1}{n} \sum_{i=1}^{n} \log \frac{p(x_i|\theta_0)}{\sum_{k=1}^{K} \overline{a}_k c(x_i|\overline{b}_k) \exp\{-\frac{C'}{n_k + \min\{\phi_0, \xi_0\}}\}},$$

and $C'$ is a constant. Thus, from (20), evaluating the right-hand sides of (25) and (27) at specific points near the true parameter $\theta_0$, we obtain the upper bound in (23). The lower bound in (23) is obtained from (25) and (27) by Jensen's inequality and the constraint $\sum_{k=1}^{K} \overline{a}_k = 1$. **(Q.E.D)**

## 6   Discussion and Conclusion

In this paper, we showed the upper and lower bounds of the stochastic complexity for the mixture of exponential family models in the VB learning.

Firstly, we compare the stochastic complexity shown in Theorem 2 with the one in the true Bayesian learning. On the mixture models with $M$ parameters in each component, the following upper bound for the coefficient of $F(n)$ in (10) is known [13],

$$\lambda \leq \begin{cases} (K + K_0 - 1)/2 & (M = 1), \\ (K - K_0) + (MK_0 + K_0 - 1)/2 & (M \geq 2). \end{cases} \tag{28}$$

By the certain conditions about the prior distribution under which the above bound was derived, we can compare the stochastic complexity when $\phi_0 = 1$. Putting $\phi_0 = 1$ in (24), we have

$$\overline{\lambda} = K - K_0 + (MK_0 + K_0 - 1)/2. \tag{29}$$

Since we obtain $\overline{F}(n) \simeq \overline{\lambda} \log n + O(1)$ under certain assumptions[11], let us compare $\overline{\lambda}$ of the VB learning to $\lambda$ in (28) of the true Bayesian learning. When $M = 1$, that is, each component has one parameter, $\overline{\lambda} \geq \lambda$ holds since $K_0 \leq K$. This means that the more redundant components the model has, the more the VB learning differs from the true Bayesian learning. In this case, $2\overline{\lambda}$ is equal to the number of the parameters of the model. Hence the BIC[9] corresponds to $\overline{\lambda} \log n$ when $M = 1$. If $M \geq 2$, the upper bound of $\lambda$ is equal to $\overline{\lambda}$. This implies that the variational posterior is close to the true Bayesian posterior when $M \geq 2$. More precise discussion about the accuracy of the approximation can be done for models on which tighter bounds or exact values of the coefficient $\lambda$ in (10) are given[10].

Secondly, we point out that Theorem 2 shows how the hyperparameter $\phi_0$ influence the process of the VB learning. The coefficient $\overline{\lambda}$ in (24) indicates that only when $\phi_0 \leq (M + 1)/2$, the prior distribution (2) works to eliminate the redundant components that the model has and otherwise it works to use all the components.

And lastly, let us give examples of how to use the theoretical bounds in (23). One can examine experimentally whether the actual iterative algorithm converges to the optimal variational posterior instead of local minima by comparing the stochastic complexity with our theoretical result. The theoretical bounds would also enable us to compare the accuracy of the VB learning with that of the Laplace approximation or the MCMC method. As mentioned in Section 4, our result will be important for developing effective model selection methods using $\overline{F}(X^n)$ in the future work.

## Footnotes

[1]$K(q(x)||p(x))$ denotes the Kullback information from a distribution $q(x)$ to a distribution $p(x)$, that is,

[2]$\langle \cdot \rangle_{p(x)}$ denotes the expectation over $p(x)$.

[3]Hereafter, we omit the condition $X^n$ of the variational posteriors, and abbreviate them to $q(Y^n, \theta)$, $Q(Y^n)$ and $r(\theta)$.

[4] $\frac{\partial^2 g(b)}{\partial b \partial b}$ denotes the matrix whose $ij$th entry is $\frac{\partial^2 g(b)}{\partial b^{(i)} \partial b^{(j)}}$ and $|\cdot|$ denotes the determinant of a matrix.

[5] $O_p(1)$ denotes a random variable bounded in probability.

# References

[1] H.Akaike, "Likelihood and Bayes procedure," *Bayesian Statistics*, (Bernald J.M. eds.) University Press, Valencia, Spain, pp.143-166, 1980.

[2] H.Attias, "Inferring parameters and structure of latent variable models by variational bayes," *Proc. of UAI*, 1999.

[3] L.D.Brown, "Fundamentals of statistical exponential families," IMS Lecture Notes-Monograph Series, 1986.

[4] Z.Ghahramani, M.J.Beal, "Graphical models and variational methods," *Advanced Mean Field Methods* , MIT Press, 2000.

[5] J.A.Hartigan, "A Failure of likelihood asymptotics for normal mixtures," *Proc. of the Berkeley Conference in Honor of J.Neyman and J.Kiefer*, Vol.2, 807-810, 1985.

[6] D.J. Mackay, "Bayesian interpolation," *Neural Computation*, 4(2), pp.415-447, 1992.

[7] G.McLachlan, D.Peel,"Finite mixture models," Wiley, 2000.

[8] M.Sato, "Online model selection based on the variational bayes," *Neural Computation*, 13(7), pp.1649-1681, 2001.

[9] G.Schwarz, "Estimating the dimension of a model," *Annals of Statistics*, 6(2), pp.461-464, 1978.

[10] K.Watanabe, S.Watanabe, "Lower bounds of stochastic complexities in variational bayes learning of gaussian mixture models," *Proc. of IEEE CIS04*, pp.99-104, 2004.

[11] K.Watanabe, S.Watanabe, "Stochastic complexity for mixture of exponential families in variational bayes," *Proc. of ALT05*, pp.107-121, 2005.

[12] S.Watanabe,"Algebraic analysis for non-identifiable learning machines," *Neural Computation*, 13(4), pp.899-933, 2001.

[13] K.Yamazaki, S.Watanabe, "Singularities in mixture models and upper bounds of stochastic complexity," *Neural Networks*, 16, pp.1029-1038, 2003.
